# CONSTRAINTS ON ADAPTIVE NETWORKS FOR MODELING HUMAN GENERALIZATION

M. Pavel                     Mark A. Gluck                     Van Henkle

*Department of Psychology*
*Stanford University*
*Stanford, CA 94305*

## ABSTRACT

The potential of adaptive networks to learn categorization rules and to model human performance is studied by comparing how natural and artificial systems respond to new inputs, i.e., how they generalize. Like humans, networks can learn a deterministic categorization task by a variety of alternative individual solutions. An analysis of the constraints imposed by using networks with the minimal number of hidden units shows that this "minimal configuration" constraint is not sufficient to explain and predict human performance; only a few solutions were found to be shared by both humans and minimal adaptive networks. A further analysis of human and network generalizations indicates that initial conditions may provide important constraints on generalization. A new technique, which we call "reversed learning", is described for finding appropriate initial conditions.

## INTRODUCTION

We are investigating the potential of adaptive networks to learn categorization tasks and to model human performance. In particular we have studied how both natural and artificial systems respond to new inputs, that is, how they **generalize**. In this paper we first describe a computational technique to analyze generalizations by adaptive networks. For a given network structure and a given classification problem, the technique enumerates all possible network solutions to the problem. We then report the results of an empirical study of human categorization learning. The generalizations of human subjects are compared to those of adaptive networks. A cluster analysis of both human and network generalizations indicates significant differences between human performance and possible network behaviors. Finally, we examine the role of the initial state of a network for biasing the solutions found by the network. Using data on the relations between human subjects' initial and final performance during training, we develop a new technique, called "reversed learning", which shows some potential for modeling human learning processes using adaptive networks. The scope of our analyses is limited to generalizations in deterministic pattern classification (categorization) tasks.

The basic difficulty in generalization is that there exist many different classification rules ("solutions") that that correctly classify the training set but which categorize novel objects differently. The number and diversity of possible solutions depend on the language defining the pattern recognizer. However, additional constraints can be used in conjunction with many types of pattern categorizers to eliminate some, hopefully undesirable, solutions.

One typical way of introducing additional constraints is to minimize the representation. For example minimizing the number of equations and parameters in a mathematical expression, or the number of rules in a rule-based system would assure that some identification maps would not be computable. In the case of adaptive networks, minimizing the size of adaptive networks, which reduces the number of possible encoded functions, may result in improved generalization performance (Rumelhart, 1988).

The critical theoretical and applied questions in pattern recognition involve characterization and implementation of desirable constraints. In the first part of this paper we describe an analysis of adaptive networks that characterizes the solution space for any particular problem.

## ANALYSES OF ADAPTIVE NETWORKS

Feed-forward adaptive networks considered in this paper will be defined as directed graphs with linear threshold units (LTU) as nodes and with edges labeled by real-valued weights. The output or activations of a unit is determined by a monotonic nonlinear function of a weighted sum of the activation of all units whose edges terminate on that unit. There are three types of units within a feed-forward layered architecture: (1) Input units whose activity is determined by external input; (2) output units whose activity is taken as the response; and (3) the remaining units, called hidden units. For the sake of simplicity our discussion will be limited to objects represented by binary valued vectors.

A fully connected feed-forward network with an unlimited number of hidden units can compute any boolean function. Such a general network, therefore, provides no constraints on the solutions. Therefore, additional constraints must be imposed for the network to prefer one generalization over another. One such constraint is minimizing the size of the network. In order to explore the effect of minimizing the number of hidden units we first identify the minimal network architecture and then examine its generalizations.

Most of the results in this area have been limited to finding bounds on the expected number of possible patterns that could be classified by a given network (e.g. Cover, 1965; Volper and Hampson, 1987; Valiant, 1984; Baum & Haussler, 1989). The bounds found by these researchers hold for all possible categorizations and are, therefore, too broad to be useful for the analysis of particular categorization problems.

To determine the generalization behavior for a particular network architecture, a specific

categorization problem and a training set it is necessary to find find all possible solutions and the corresponding generalizations. To do this we used a computational (not a simulation) procedure developed by Pavel and Moore (1988) for finding minimal networks solving specific categorization problems. Pavel and Moore (1988) defined two network solutions to be different if at least one hidden unit categorized at least one object in the training set differently. Using this definition their algorithm finds all possible different solutions. Because finding network solutions is NP-complete (Judd, 1987), for larger problems Pavel and Moore used a probabilistic version of the algorithm to estimate the distribution of generalization responses.

One way to characterize the constraints on generalization is in terms of the number of possible solutions. A larger number of possible solutions indicates that generalizations will be less predictable. The critical result of the analysis is that, even for minimal networks, the number of different network solutions is often quite large. Moreover, the number of solutions increases rapidly with increases in the number of hidden units. The apparent lack of constraints can also be demonstrated by finding the probability that a network with a randomly selected hidden layer can solve a given categorization problem. That is, suppose that we select *n different* hidden units, each unit representing a linear discriminant function. The activations of these random hidden units can be viewed as a transformation of the input patterns. We can ask what is the probability that an output unit can be found to perform the desired dichotomization. A typical example of a result of this analysis is shown in Figure 1 for the three-dimensional (3D) parity problem. In the minimal configuration involving three hidden units there were 62 different solutions to the 3D parity problem. The rapid increase in probability (high slope of the curve in Figure 1) indicates that adding a few more hidden units rapidly increases the probability that a random hidden layer will solve the 3D parity problem.

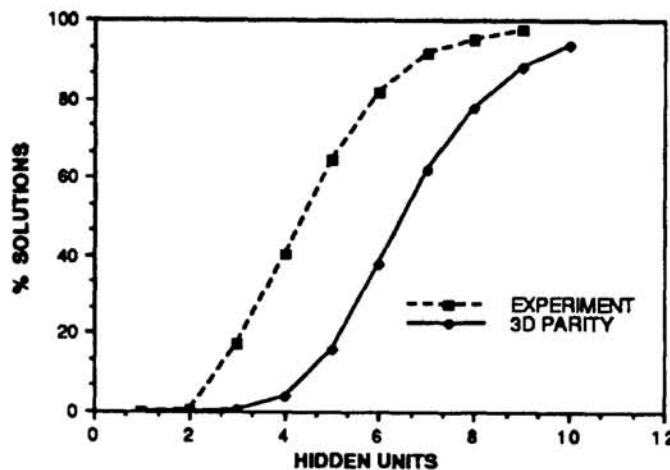

*Figure 1* The proportion of solutions to 3D parity problem (solid line) and the experimental task (dashed line) as a function of the number of hidden units.

The results of a more detailed analysis of the generalization performance of the minimal networks will be discussed following a description of a categorization experiment with

human subjects.

# HUMAN CATEGORIZATION EXPERIMENT

In this experiment human subjects learned to categorize objects which were defined by four dimensional binary vectors. Of the $2^4$ possible objects, subjects were trained to classify a subset of 8 objects into two categories of 4 objects each. The specific assignments of objects into categories was patterned after Medin et al. (1982) and is shown in Figure 2. Eight of the patterns are designated as a training set and the remaining eight comprise the test set. The assignment of the patterns in the training set into two categories was such that there were many combinations of rules that could be used to correctly perform the categorization. For example, the first two dimensions could be used with one other dimension. The training patterns could also be categorized on the basis of an *exclusive or (XOR)* of the last two dimensions. The type of solution obtained by a human subject could only be determined by examining responses to the test set as well as the training set.

| | | TRAINING SET | | TEST SET | |
|---|---|---|---|---|---|
| | $X_1$ | 1 1 0 1 | 0 0 1 0 | 0 0 0 1 | 1 1 0 1 |
| DIMENSIONS | $X_2$ | 1 1 1 0 | 0 0 0 1 | 0 0 1 0 | 1 1 1 0 |
| | $X_3$ | 1 0 1 0 | 1 0 1 0 | 0 1 0 1 | 0 1 0 1 |
| | $X_4$ | 1 0 1 0 | 0 1 0 1 | 0 1 0 1 | 0 1 0 1 |
| CATEGORY | | A A A A | B B B B | ? ? ? ? | ? ? ? ? |

*Figure 2.* Patterns to be classified. (Adapted from Medin et al., 1982).

In the actual experiments, subjects were asked to perform a medical diagnosis for each pattern of four symptoms (dimensions). The experimental procedure will be described here only briefly because the details of this experiment have been described elsewhere in detail (Pavel, Gluck, Henkle, 1988). Each of the patterns was presented serially in a randomized order. Subjects responded with one of the categories and then received feedback. The training of each individual continued until he reached a criterion (responding correctly to 32 consecutive stimuli) or until each pattern had been presented 32 times. The data reported here is based on 78 subjects, half (39) who learned the task to criterion and half who did not.

Following the training phase, subjects were tested using all 16 possible patterns. The results of the test phase enabled us to determine the generalizations performed by the subjects. Subjects' generalizations were used to estimate the "functions" that they may have been using. For example, of the 39 criterion subjects, 15 used a solution that was consistent with the exclusive-or (XOR) of the dimensions $x_3$ and $x_4$.

We use "response profiles" to graph responses for an ensemble of functions, in this case for a group of subjects. A response profile represents the probability of assigning each

pattern to category "A". For example, the response profile for the XOR solution is shown in Figure 3A. For convenience we define the responses to the test set as the "generalization profile". The response profile of all subjects who reached the criterion is shown in Figure 3B. The responses of our criterion subjects to the training set were basically identical and correct. The distribution of subjects' generalization profiles reflected in the overall generalization profile are indicative of considerable individual differences

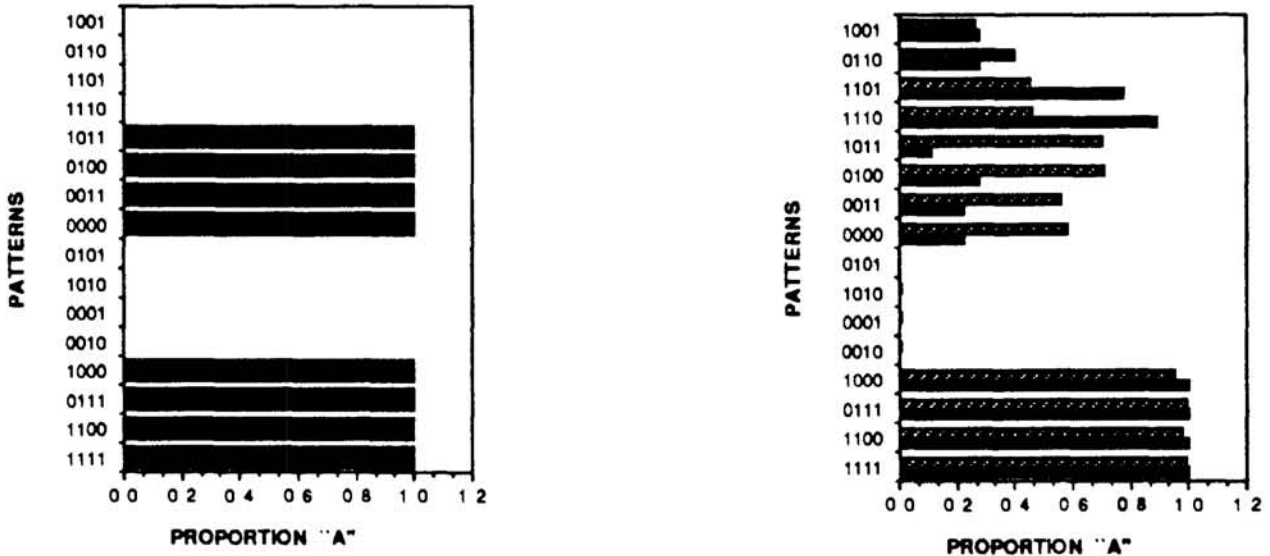

*Figure 3.* (A) Response profile of the XOR solution, and (B) a proportion of the response "A" to all patterns for human subjects (dark bars) and minimal networks (light bars). The lower 8 patterns are from the training set and the upper 8 patterns from the test set.

## MODELING THE RESPONSE PROFILE

One of our goals is to model subjects' distribution of categorizations as represented by the response profile in Figure 3B. We considered three natural approaches to such modeling: (1) Statistical/proximity models, (2) Minimal disjunctive normal forms (DNF), and (3) Minimal two-layer networks.

The statistical approach is based on the assumption that the response profile over subjects represents the probability of categorizations performed by each subject. Our data are not consistent with that assumption because each subject appeared to behave deterministically. The second approach, using the minimal DNF is also not a good candidate because there are only four such solutions and the response profile over those solutions differs considerably from that of the subjects. Turning to the adaptive network solutions, we found all the solutions using the linear programming technique described above (Pavel & Moore, 1988). The minimal two-layer adaptive network that was capable of solving the training set problem consisted of two hidden units. The proportion of solutions as a

function of the number of hidden units is shown in Figure 1 by the dashed line.

For the minimal network there were 18 different solutions. These 18 solutions had 8 different individual generalization profiles. Assuming that each of the 18 network solution is equally likely, we computed the generalization profile for minimal network shown in Figure 3B. The response profile for the minimal network represents the probability that a randomly selected minimal network will assign a given pattern to category "A". Even without statistical testing we can conclude that the generalization profiles for humans and networks are quite different. It is possible, however, that humans and minimal networks obtain similar solutions and that the differences in the average responses are due to the particular statistical sampling assumption used for the minimal networks (i.e. each solution is equally likely). In order to determine the overlap of solutions we examined the generalization profiles in more detail.

*CLUSTERING ANALYSIS OF GENERALIZATION PROFILES*

To analyze the similarity in solutions we defined a metric on generalization profiles. The Hamming distance between two profiles is equal to the number of patterns that are categorized differently. For example, the distance between generalization profile "A A B A B B B B" and "A A B B B B A B" is equal to two, because the two profiles differ on only the fourth and seventh pattern. Figure 4 shows the results of a cluster analysis using a hierarchical clustering procedure that maximizes the average distance between clusters.

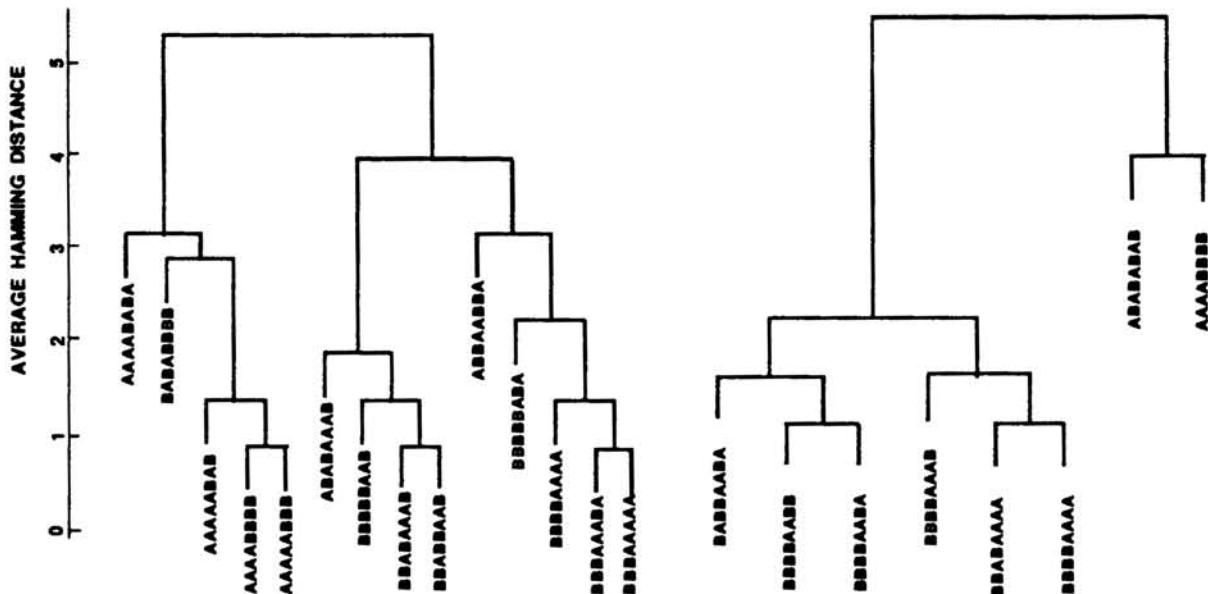

*Figure 4*. Results of hierarchical clustering for human (left) and network (right) generalization profiles.

In this graph the average distance between any two clusters is shown by the value of the lowest common node in the tree. The clustering analysis indicates that humans and

networks obtained widely different generalization profiles. Only three generalization profiles were found to be common to human and networks. This number of common generalizations is to be expected by chance if the human and network solutions are independent. Thus, even if there exists a learning algorithm that approximates the human probability distribution of responses, the minimal network would not be a good model of human performance in this task.

It is clear from the previously described network analysis that somewhat larger networks with different constraints could account for human solutions. In order to characterize the additional constraints, we examined subjects' individual strategies to find out why individual subjects obtained different solutions.

## ANALYSIS OF HUMAN LEARNING STRATEGIES

Human learning strategies that lead to preferences for particular solutions may best be modeled in networks by imposing constraints and providing hints (Abu-Mostafa 1989). These include choosing the network architecture and a learning rule, constraining connectivity, and specifying initial conditions. We will focus on the specification of initial conditions.

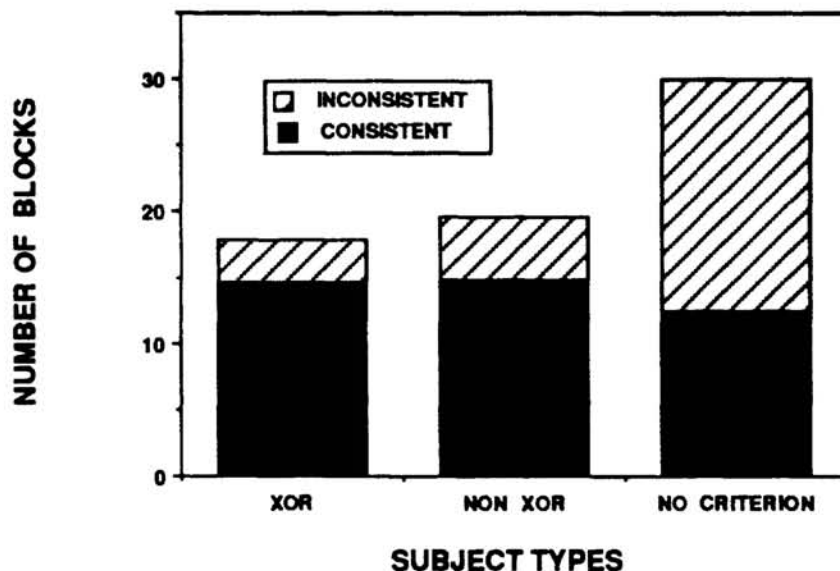

*Figure 5.* The number of consistent or non-stable responses (black) and the number of stable incorrect responses (light) for XOR, Non-XOR criterion subjects, and for those who never reached criterion.

Our effort to examine initial conditions was motivated by large differences in learning curves (Pavel et al., 1988) between subjects who obtained the XOR solutions and those who did not. The subjects who did not obtain the XOR solutions would perform much better on some patterns (e.g. 0001) then the XOR subjects, but worse on other patterns (e.g. 1000). We concluded that these subjects during the first few trials discovered rules

that categorized most of the training patterns correctly but failed on one or two training patterns.

We examined the sequences of subjects' responses to see how well they adhered to "incorrect" rules. We designated a response to a pattern as *stable* if the individual responded the same way to that pattern at least four times in a row. We designated a response as *consistent* if the response was stable and correct. The results of the analysis are shown in Figure 5. These results indicate that the subjects who eventually achieved the XOR solution were less likely to generate stable incorrect solutions. Another important result is that those subjects who never learned the correct responses to the training set were not responding randomly. Rather, they were systematically using incorrect rules. On the basis of these results, we conclude that subjects' initial strategies may be important determinants of their final solutions.

*REVERSED LEARNING*

For simplicity we identify subjects' initial conditions by their responses on the first few trials. An important theoretical question is whether or not it is possible to find a network structure, initial conditions and a learning rule such that the network can represent both the initial and final behavior of the subject. In order to study this problem we developed a technique we call "reversed learning". It is based on a perturbation analysis of feed-forward networks. We use the fact that the error surface in a small neighborhood of a minimum is well approximated by a quadratic surface. Hence, a well behaved gradient descent procedure with a starting point in the neighborhood of the minimum will find that minimum.

The reversed learning procedure consists of three phases. (1) A network is trained to a final desired state of a particular individual, using both the training and the test patterns. (2) Using only the training patterns, the network is then trained to achieve the initial state of that individual subject closest to the desired final state (3) The network is trained with only the training patterns and the solution is compared to the subject's response profiles. Our preliminary results indicate that this procedure leads in many cases to initial conditions that favor the desired solutions. We are currently investigating conditions for finding the optimal initial states.

## CONCLUSION

The main goal of this study was to examine constraints imposed by humans (experimentally) and networks (linear programming) on learning of simple binary categorization tasks. We characterize the constraints by analyzing responses to novel stimuli. We showed that, like the humans, networks learn the deterministic categorization task and find many, very different, individual solutions. Thus adaptive networks are better models than statistical models and DNF rules. The constraints imposed by minimal networks, however, appear to differ from those imposed by human learners in that there are only a few solutions shared between human and adaptive networks. After a detailed analysis of

the human learning process we concluded that initial conditions may provide important constraints. In fact we consider the set of initial conditions as powerful "hints" (Abu-Mostafa, 1989) which reduces the number of potential solutions, without reducing the complexity of the problem. We demonstrated the potential effectiveness of these constraints using a perturbation technique, which we call reversed learning, for finding appropriate initial conditions.

## Acknowledgements

This work was supported by research grants from the National Science Foundation (BNS-86-18049) to Gordon Bower and Mark Gluck, and (IST-8511589) to M. Pavel, and by a grant from NASA Ames (NCC 2-269) to Stanford University. We thank Steve Sloman and Bob Rehder for useful discussions and their comments on this draft.

## References

Abu-Mostafa, Y. S. *Learning by example with hints*. NIPS, 1989.

Baum, E. B., & Haussler, D. *What size net gives valid generalization?*. NIPS, 1989.

Cover, T. (June 1965). Geometrical and statistical properties of systems of linear inequalities with applications in pattern recognition. *IEEE Transactions on Electronic Computers, EC-14*, 3, 326-334.

Judd, J. S. *Complexity of connectionist learning with various node functions*. Presented at the First IEEE International Conference on Neural Networks, San Diego, June 1987.

Medin, D. L., Altom, M. W., Edelson, S. M., & Freko, D. (1982). Correlated symptoms and simulated medical classification. *Journal of Experimental Psychology: Learning, Memory, & Cognition, 8(1)*, 37-50.

Pavel, M., Gluck, M. A., & Henkle, V. *Generalization by humans and multi-layer adaptive networks*. Submitted to *Tenth Annual Conference of the Cognitive Science Society*, August 17-19, 1988.

Pavel, M., & Moore, R. T. (1988). *Computational analysis of solutions of two-layer adaptive networks*. APL Technical Report, Dept. of Psychology, Stanford University.

Valiant, L. G. (1984). A theory of the learnable. *Comm. ACM, 27*, 11, 1134-1142.

Volper, D. J., & Hampson, S. E. (1987). Learning and using specific instances. *Biological Cybernetics, 56*, .